# Global Distortions from Local Rewards: Neural Coding Strategies in Path-Integrating Neural Systems

**Francisco Acosta**
Department of Physics
UC Santa Barbara
facosta@ucsb.edu

**Fatih Dinc**
CNC Program
Stanford University

**William T. Redman**
Applied Physics Laboratory
Johns Hopkins University

**Manu Madhav**
Biomedical Engineering
University of British Columbia

**David Klindt**
NeuroAI
CSHL

**Nina Miolane**
Electrical & Computer Engineering
UC Santa Barbara

## Abstract

Grid cells in the mammalian brain are fundamental to spatial navigation, and therefore crucial to how animals perceive and interact with their environment. Traditionally, grid cells are thought support path integration through highly symmetric hexagonal lattice firing patterns. However, recent findings show that their firing patterns become distorted in the presence of significant spatial landmarks such as rewarded locations. This introduces a novel perspective of dynamic, subjective, and action-relevant interactions between spatial representations and environmental cues. Here, we propose a practical and theoretical framework to quantify and explain these interactions. To this end, we train path-integrating recurrent neural networks (piRNNs) on a spatial navigation task, whose goal is to predict the agent's position with a special focus on rewarded locations. Grid-like neurons naturally emerge from the training of piRNNs, which allows us to investigate how the two aspects of the task, space and reward, are integrated in their firing patterns. We find that geometry, but not topology, of the grid cell population code becomes distorted. Surprisingly, these distortions are global in the firing patterns of the grid cells despite local changes in the reward. Our results indicate that after training with location-specific reward information, the preserved representational topology supports successful path integration, whereas the emergent heterogeneity in individual responses due to global distortions may encode dynamically changing environmental cues. By bridging the gap between computational models and the biological reality of spatial navigation under reward information, we offer new insights into how neural systems prioritize environmental landmarks in their spatial navigation code.

## 1 Introduction

**Motivation** Neuroscientists describe computations in the brain by jointly considering the firing activity of a population of $N$ neurons in *neural state space* $\mathbb{R}^N$. In this space, each basis vector corresponds to one neuron, one point summarizes the activity of all neurons at a given instant, and a trajectory in neural state space describes the time evolution of the firing of all neurons. Across many brain regions and modalities, these trajectories have been observed to be mostly constrained to a lower dimensional manifold within the state space: the *neural manifold* (see Figure 1 right). Such manifolds have been observed in the motor cortex [1, 2], the hippocampus [3, 4], visual cortex [5, 6], head direction circuit [7, 8], and grid cells in the entorhinal cortex [9]. Characterizing the geometric properties of neural manifolds including their dimension, topology, and curvature enables a wholistic understanding of multi-neuron activity through time, and a quantified approach to describe computations in the brain [10, 11, 12].

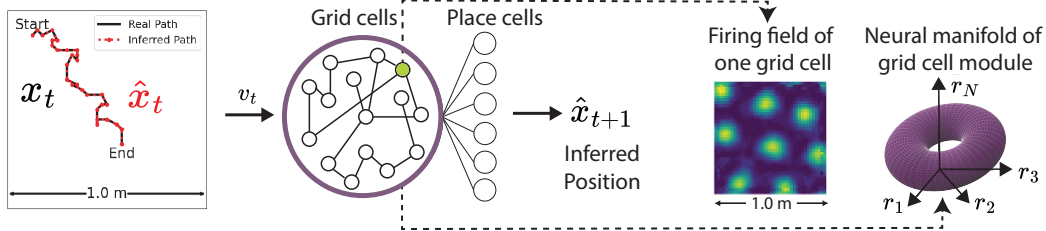

Figure 1: **Path integrating Neural Systems.** Left: an artificial agent explores its 2D environment traveling along the path $x_t$ shown in black. Middle: the agent's velocity $v_t$ is given as input to neurons in a path-integrating recurrent neural network (piRNN) or grid cells in the mammalian brain [13]. The agent maintains a representation of its movement across multiple grid cell modules. This representation is linearly decoded onto place cells, providing an estimate of the agent's new position $\hat{x}_{t+1}$. Right: Each grid cell has a firing field that is a hexagonal lattice. Together, grid cells' activity within a grid cell module forms a 2D torus [9, 20].

Grid cells are neurons in the medial entorhinal cortex (MEC) which exhibit activity patterns characterized by hexagonal lattices across space [13] (see firing field on Figure 1). Grid cells can be grouped into distinct subpopulations called *modules*, specified by the spatial period of their hexagonal lattice [14]. The neural manifold corresponding to neurons in a grid cell module has been shown to form a 2D torus in the high-dimensional state space [9]. The traditional view in neuroscience is that grid cells support *path integration* —the process used by an animal to keep track of its position by integrating its past position and current speed —by providing a global metric for physical space. However, recent experiments suggest that grid cell lattices become distorted in the presence of relevant landmarks, such as rewards [15, 16, 17]. In other words: grid cell activity might not reflect true physical distances but instead underlies a metric for subjective action-relevant space [18]. While the distortion of grid cell activity has been reported qualitatively, it remains to be described quantitatively.

**Contributions** We propose a practical and theoretical framework to quantify and explain the distortion of grid cell neural manifolds under task-relevant reward information. We apply this framework to compare computational and biological neural systems performing spatial navigation.

Our technical contributions are:

1. We adapt the training of biologically-inspired RNNs performing path integration (piRNNs) [19] to account for spatial rewards via a new saliency loss term.

2. We introduce a methodology to quantify the deformation of grid cells' firing fields and neural manifolds in two scenarios —absence and presence of rewards and describe the topological changes in the grid cell module tori after training with reward.

3. We illustrate the implicit regularization achieved by fixing the readout weights, which retains the existing toroidal topology of the neural manifolds during reward training.

Our conceptual contributions are:

1. We quantify the introduction of reward in spatial navigation tasks by linking the geometric distortions of grid cells' responses and corresponding distortions of their neural manifolds.

2. In piRNNs, we link the emergent distortions of the individual hexagonal responses to geometric deformations of the neural manifold.

3. We provide evidence suggesting that grid cells may be involved in continual learning.

After reward training, the toroidal topology continue support successful path integration, while geometric deformations of the neural —corresponding to global distortions in the inidividual responses —may represent local, dynamic environmental cues, such as reward information. This dual representation allows the grid cell modules to maintain their foundational navigation capabilities while adapting to new environmental variables.

## 2    Background & Related Works

### 2.1    Neuroscience of Spatial Navigation

**Path integration** is the process by which animals use self-motion cues including information from the vestibular system and proprioception to integrate their past movements and positions, allowing them to maintain an estimate of their position. Theoretical models have explored mechanisms by which path integration can be performed by neural networks in the mammalian brain [21], most notably through networks of place cells and grid cells [22].

**Place cells** are a functional class of neurons found in the hippocampus [23], a region of the brain implicated in memory and spatial navigation. The neural activity of place cells is believed to represent the animal's current physical location in space, thereby creating a "cognitive map" [24].

**Grid cells** are a functional class of neurons found in the medial entorhinal cortex [13] whose spatial firing pattern form a hexagonal lattice covering their 2D environment in open field settings. Many works [25, 26, 27] show that grid cells play an important role in *path integration*, by integrating self-motion information sending neuronal signals to their downstream place cells.

The **grid cell firing field** can be described by three parameters: (1) *spacing*, or distance between lattice points, (2) *orientation* of the lattice relative to a reference direction, and (3) *phase*, or the lattice offset relative to a reference point. A **grid cell module** consists of grid cells with the same spacing and orientation but varying phases [14]. Each module is known to form a neural manifold with the topology of the 2D torus $\mathcal{T}^2$ [9].

### 2.2    RNNs for Spatial Navigation

Biologically-inspired, computational models of place cells and grid cells have emerged in recent years. Specifically, recurrent neural networks trained to perform path integration (piRNNs) of artificial agents have been shown to exhibit artificial neurons that behave like biological grid cells [28, 29, 30, 19, 31, 32]. This paradigm allows for systematic exploration of phenomena observed in neural circuits involved in spatial navigation, such as hippocampal remapping [33]. Below, we introduce the piRNN framework of spatial navigation from [19].

Let $x \in \mathbb{R}^2$ denote the agent's position, $\Delta x \in \mathbb{R}^2$ the displacement in unit time, $v$ the velocity, $r(x) \in \mathbb{R}^N$ be the activity of $N$ grid cells, $\phi(x) \in \mathbb{R}^P$ be the activity of $P$ place cells; $r_i(x)$ and $\phi_p(x)$ being the activity of the $i^{\text{th}}$ grid cell and $p^{\text{th}}$ place cell, respectively —see Figure 1 (left). The grid cell activity $r(x)$ for each module forms a toroidal *neural manifold* embedded in the *neural state space* $\mathbb{R}^N$: see torus on Figure 1 (right).

**Place cells in piRNNs:**    As done in the literature [30], we model place cell activity $\phi$ as a linear readout of the activities, $r$, of the piRNN neurons. The position estimate $\hat{x}$ is decoded from $\phi$ as:

$$\hat{\phi}^t = Qr^t, \ \ \hat{x}_t = \underset{p \in P}{\arg\max} \, \hat{\phi}_p^t, \tag{1}$$

where $Q \in \mathbb{R}^{P \times N}$ is the place cell readout matrix. The architecture is illustrated in Figure 1. Previous work has shown that the piRNN neurons, $r$, become grid cells after training [30].

**Path integration by piRNNs:**    The piRNN is trained to perform path-integration and correctly infer the spatial position. Its goal is to learn the place cell readout matrix $Q$, the neural representation $(r(x), \forall x)$, its recurrent weight matrix $W \in \mathbb{R}^{N \times N}$, and its input weight matrix $U \in mathbbR^{2 \times N}$ via minimization of the loss function $\mathcal{L}$. The loss $\mathcal{L}$ used by Xu et al. [19] is:

$$\mathcal{L} = \mathcal{L}_{\text{error}} + \mathcal{L}_{\text{kernel}} + \mathcal{L}_{\text{conformal}} + \mathcal{L}_{\text{regularization}}. \tag{2}$$

The error on the prediction of the spatial position is written as $\mathcal{L}_{\text{error}} = \mathbb{E}_{x, \Delta x} [L_{\text{error}}]$ where:

$$L_{\text{error}} = \sum_{t=1}^{T} \|\phi \left( x + \Delta x_1 + \ldots + \Delta x_t \right) - Qr(x + \Delta x_1 + \ldots + \Delta x_t)\|^2. \tag{3}$$

We refer to Sec. S3 for details on the other terms: $\mathcal{L}_{\text{kernel}}, \mathcal{L}_{\text{conformal}}, \mathcal{L}_{\text{regularization}}$.

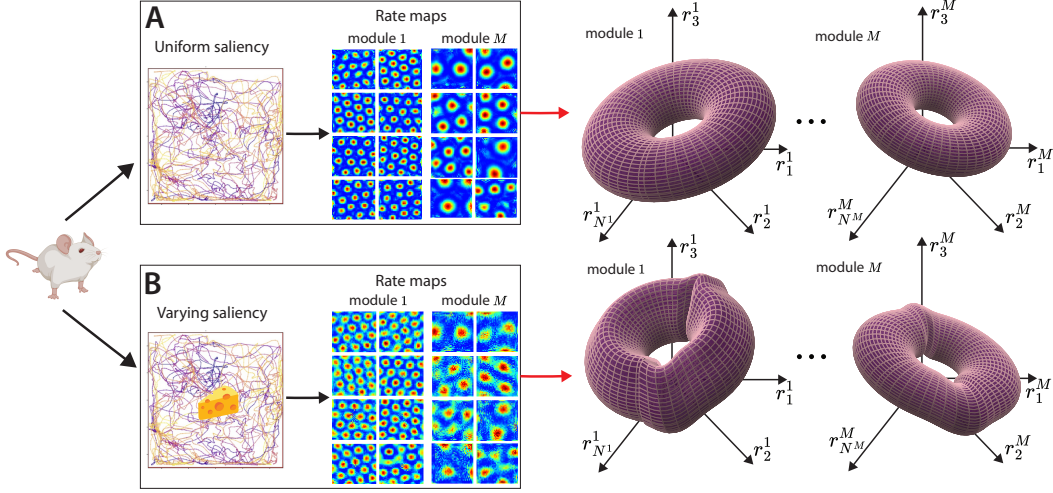

Figure 2: **Geometry of grid cell module tori is changed by presence of salient features in the environment.**
**A.** An agent (for us, a piRNN) is trained to perform path integration in its 2D environment with uniform spatial saliency. Canonical grid cells develop hexagonal lattice responses (rate maps) across $M$ modules. The population activity of a single grid cell module forms a torus in neural state space. **B.** The same agent undergoes a second phase of training, with its environment now containing rewards (areas of high importance, or *saliency*). We model this saliency by modifying the loss of our piRNN to prioritize accurate position decoding near rewards. Its grid cells adjust their individual responses, which we link to geometric deformations of the neural tori.

## 3 Theory of Deformations: From Firing Fields to Neural Manifolds

Recent experimental evidence in animals and our own experiments on piRNNs reveal that grid cell firing fields get distorted after introducing a reward in the agent's spatial environment. Here, we propose a theory to relate deformations of the firing fields (see Figure 1 middle) to deformations of the corresponding neural manifolds (see Figure 1 right) during path-integration with spatially localized rewards. We first present the general theory of deformations. Then, we illustrate it with two scenarios observed in practice: (i) smoothing of firing fields (diffused units) observed in our piRNN experiments as described in Section 4, and (ii) attraction of firing fields (attracted units) observed in [15].

### 3.1 General Theory

**Deformation of firing fields**   Consider one grid cell $i$ with firing field $x \mapsto r_i(x)$ over the environment $\mathbb{R}^2$ (Figure 1 middle). In line with experimental observations, we assume that the introduction of a reward, modeled by the *saliency map* $s(x)$, deforms the firing field of the grid cell (see Figure 2). We model this deformation as a diffeomorphism $\Phi : \mathbb{R}^2 \mapsto \mathbb{R}^2$ of the environment.

We further assume that every grid cell $i$ has a constant *firing energy budget*, in that: $\int_{x \in \mathbb{R}^2} r_i(x)dx$ is constant. Putting this together, introducing a reward in the environment yields a new firing field for grid cell $i$ written as:

$$r_i^{\Phi}(x) = |\det \Phi'(x)|.r_i \circ \Phi(x) \in \mathbb{R} \text{ for all } x \in \mathbb{R}^2, \tag{4}$$

where $\Phi'$ is the Jacobian of $\Phi$, $|\det \Phi'(x)|$ represents a change in volume induced by the deformation $\Phi$, and "." denotes scalar multiplication. This formulation deforms the shapes of the firing fields, and modifies their amplitudes in order to keep the firing energy budget constant. This statement is made precise in the proposition below.

**Proposition 1.** *The deformation of the grid cell firing fields $r_i$ by $\Phi$ given by $r_i * \Phi = r_i^{\Phi}$ defines a right group action of the group of diffeomorphisms over the space of firing fields. The firing energy budget is an invariant of this group action.*

The definition of group action and the proof are given in Appendix S1. Here, we do not specify a formula for $\Phi$. Instead, we suppose that $\Phi$ can be estimated from experiments with piRNNs or animals performing path-integration tasks without and with rewards.

**Deformation of neural manifold**  The deformation of each firing field via $\Phi$ induces a deformation of the associated neural manifold: the torus associated to a given grid cell module changes shape in neural state space $\mathbb{R}^N$ (see Figure 2 right). Before deformation, the torus neural manifold can be represented by the parameterized surface $x \mapsto R(x) = [r_1(x), ..., r_N(x)]^T$, where the periodicity of the firing fields means that the function $R$ is not necessarily injective: one point on the manifold surface may correspond to several locations $x$'s. Introducing a reward in the environment yields a new neural manifold written as:

$$R^\Phi(x) = |\det \Phi'(x)|.R \circ \Phi(x) \in \mathbb{R}^N \text{ for all } x \in \mathbb{R}^2. \tag{5}$$

In what follows, we consider that $R$ is injective, or we restrict $R$ to its domain of injectivity. In this context, we can predict some geometric properties of the deformed neural manifold, as given in the proposition below.

**Proposition 2.** *The deformation of the neural manifold $R$ by $\Phi$ given by $R * \Phi = R^\Phi$ defines a right group action of the group of diffeomorphisms over the space of neural manifolds. The barycenter and the topology of the manifold are invariants of this group action, provided that the firing fields are regular.*

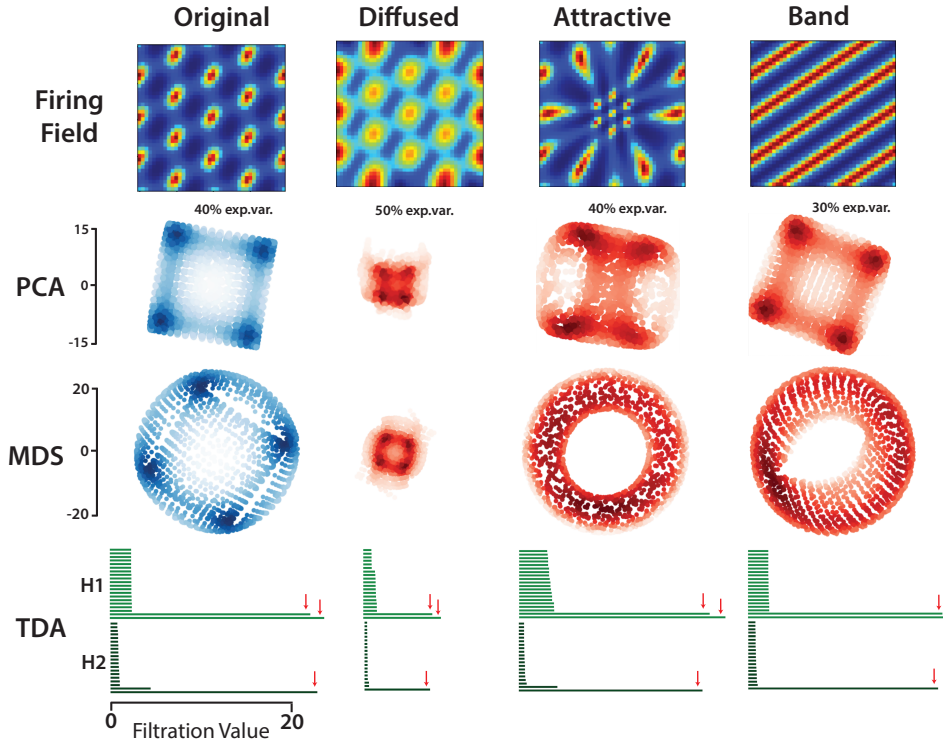

Figure 3: **Relating firing fields to neural manifolds across deformations in synthetic grid cells.** We investigate how deformations away from perfect hexagonal symmetry in the firing fields of synthetic grid cells affects the geometry of the toroidal neural manifold. From left to right. **Original units.** The original hexagonal grid cells show clear signatures of toroidal topology, as indicated by the presence of 2 loops in the first homology group (H1) and 1 void in the second homology group (H2). We show 2D projections using principal components analysis (PCA) and multidimensional scaling (MDS) to serve as baselines against which to compare the manifolds of deformed grid cells. PCA projections are consistent with a "flat" torus geometry. **Diffused units.** Diffused units created from convolution of the original grid cells with a Gaussian kernel maintain toroidal topology, but PCA and MDS show that the size of the neural manifold is reduced as predicted by theory. **Attracted units.** Inspired by experimental evidence from [15] we created attracted units from synthetic grid cells by applying a diffeomorphism to the 2D environment. While manifold size is unchanged, PCA projections suggest the torus becomes more curved in neural state space. Toroidal topology is preserved. **Band units**. We created a synthetic module with 17% of original grid units replaced with band units of same spatial scale, with uniformly distributed orientations. The geometry and topology of the resulting manifold are largely unchanged.

See Appendix S1 for the proof. Next, we consider deformations of the firing fields that have appeared empirically, either in our piRNN experiments or in animal experiments.

## 3.2 Diffused Units: Smoothing of Firing Fields

We show how smoothing of the firing fields of the grid cells, observed empirically in Section 4 impacts neural manifolds in neural state space.

**Conjecture 1** (Reduction hypothesis)**.** The smoothing of the firing fields by a Gaussian filter reduces the size $S$ of the neural manifold $R$, where the size is defined as: $S = \max_{x \in \mathbb{R}^2} \|R(x)\|$.

The size of the neural manifold becomes, after introduction of reward and deformation by $\Phi$:

$$S^{\Phi} = \max_{x \in \mathbb{R}^2} \| \int_{u \in \mathbb{R}^2} G(x-u)R(u)du\|, \tag{6}$$

where $G$ represents the Gaussian filter used for smoothing. When a function $\|R\|$ is convolved with a Gaussian filter, the resulting function is smoother than $\|R\|$. The convolution effectively averages the values of $\|R\|$ over the neighborhood defined by the Gaussian, which reduces the peaks of $\|R\|$. This is because the convolution is essentially a weighted average. Consequently, via such a deformation of the firing fields, the resulting manifold will be smaller.

**Testing the Reduction Hypothesis** In practice in our experiments, we perform principal component analysis and other dimension reduction techniques to compare the sizes of the neural manifolds before and after the deformation produced by the introduction of reward.

## 3.3 Attracted Units: Displacements of Firing Fields' Centers

We show how the attraction of the grid cells' firing field centers to reward locations, observed in experiments in [15] and simulated with exaggeration in Figure 3, impacts the neural manifold in state space.

**Conjecture 2.** Attracting the centers of the firing fields to a region of the environment expands the corresponding region of the neural manifold, provided that the firing rates are regular enough.

Indeed, the diffeomorphism $\Phi$ that represents this deformation displaces volume elements of the environment, placing more volume in the center. Therefore, the term $|\det \Phi'(x)|$ that measures volume changes will be high next to the center of the environment. The region of the neural manifold corresponding to the center of the environment will therefore be magnified.

We include a more detailed discussion on the link between rate map distortions and neural manifold geometry in Section S2.

We illustrated several synthetic grid cell modules to demonstrate how various types of common grid cell firing field deformations affect the geometry of the toroidal neural manifold, summarized in Fig. 3. Global deformations such as the emergence of diffusive units with less crisp hexagonal firing patterns, attracted units with distance-based scaling, and the addition of band-like cells to a group of grid cells did not change the toroidal topology of the modules. Moreover, as predicted by Conjecture 1, the diffused units reduced the size of the manifold. Overall, global distortions can be added to the grid modules without destroying the existing toroidal topology of the neural manifold subserving the spatial navigation. As we show in the next section, such patterns emerge as a result of the dynamic integration of environmental information during the piRNN training. This process enables the networks to simultaneously solve path integration and represent environmental cues effectively.

## 4 Empirical Results

In this section, we test the theoretical global distortions introduced in the previous section by training piRNNs to perform path integration while enforcing higher accuracy in positional decoding near salient locations to model the presence of local rewards. We first introduce the multi-phase training procedure with non-uniform saliency. Then we discuss the effects of saliency training on the toroidal topology of the neural manifold, and examine the emergent distortions in the grid cell firing fields. Finally, we confirm our theoretical prediction from the previous section regarding how diffused units lead to a reduction in manifold size.

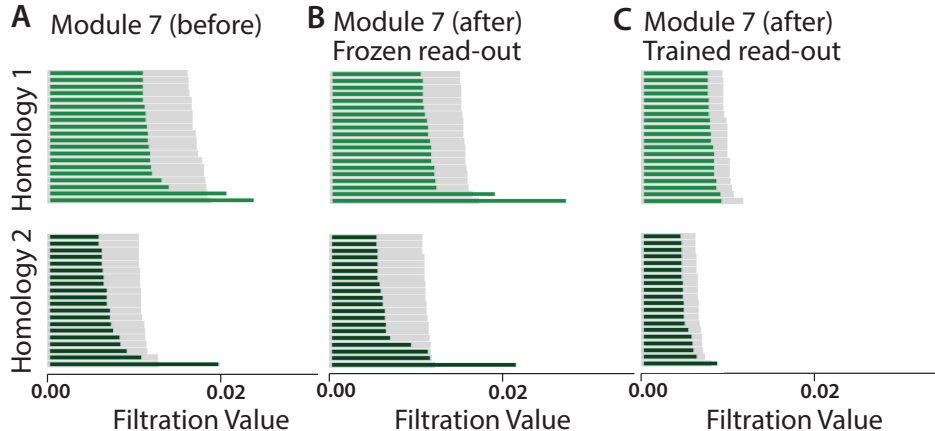

Figure 4: **Toroidal topology is preserved after saliency training when place cell read-out is frozen.** For grid cell module 7, we project the neural activity to 6 dimensions using PCA, and perform persistent homology. We show the 20 most persistent features for homology dimension 1 (loops) and homology dimension 2 (voids). Gray bars are 20 most persistent features from 10 random shuffles of the data. **A.** The population activity of module 7 of the pretrained RNN (before saliency training) has Betti numbers (1,2,1), consistent with toroidal topology. **B.** The population activity of module 7 of the RNN after saliency training retains toroidal topology, if place cell read-out is frozen (only grid representations are allowed to change). **C.** The toroidal topology is destroyed after saliency training if place cell read-out is allowed to change.

### 4.1 Training piRNNs with Non-Uniform Spatial Saliency

We investigate how the representations learned by piRNNs change through multiple phases of training incorporating spatial saliency into the objective. We begin by pretraining the piRNN model introduced in [19], which learned representations with a large fraction of units ($> 70\%$) classified as grid cells, based on the widely-used gridness score [34]. These representations exhibit a high degree of hexagonal periodicity at different scales, correspondong to distinct grid cell modules. We employed a procedure inspired by [9] to identify the grid cell modules (see Fig. S3).

After this initial pretraining phase, we modified the loss to imitate the non-uniform saliency across the environment that is characteristic of animals exploring their environments in more naturalistic settings [18] that may contain rewards. This stands in contrast to the highly uniform "open field" settings typically used in experiments, where neurons with a high degree of hexagonal periodicity are commonly found, as shown in Fig. 2. We add a kernel $(1 + s(x))$ multiplying the loss term in Eq. 3 that differentially weights the penalty of incorrect self-position estimation across different locations in the environment, thus quantifying the notion of spatial "saliency". The new loss $\mathcal{L}_{\text{error}}^s$, which modifies Eq. 3, reads

$$\mathcal{L}_{\text{error}}^s = \mathbb{E}_{x, \Delta x} \left[ (1 + s(x)) L_{\text{error}} \right]. \tag{7}$$

In our experiments, we parameterize $s(x)$ with a Gaussian of width $\sigma_*$ centered at $x_*$:

$$s(x) = s_0 \left( \exp \left( -\|x - x_*\|^2 / 2\sigma_*^2 \right) / \sqrt{2\pi\sigma_*^2} \right), \tag{8}$$

and we set $x_* = 0$ and $\sigma_* = 0.05L$, where $L$ is the length of the square environment.

### 4.2 Saliency during the training may respect or destroy the toroidal topology

We performed the second phase of training with the saliency loss in two ways. First, we allowed all connections, including the readouts from the piRNN units to the place cells, to be learned. In this case, we consistently observed a change in the topology of the grid modules, where the toroidal structure underlying the grid cells was destroyed (Fig. 4**A,C**), though the firing rates showed qualitatively similar patterns to the grid cells (Fig. S4).

As a next experiment, during the second phase of training we froze the readout weights from the RNN to the place cells and observed that the grid cell module toroidal topology was preserved (Fig. 4**A,B**). Therefore, freezing the readout may be seen as an implicit regularization that enabled continual learning in the grid cell population. By preserving the toroidal topology of the neural manifold,

the system leverages its previously learned representational structure subserving successful spatial navigation while encoding a dynamic environmental variable, *i.e.*, the saliency. Therefore, for the rest of our experiments, we fine-tuned the piRNNs with frozen readout weights.

### 4.3 Saliency-tuned piRNNs develop diverse set of distorted tuning features

In our experiments, we found that following the saliency training phase (with the modified loss term $\mathcal{L}^s_{error}$), the piRNNs developed neurons with a more heterogeneous set of spatial tuning properties, as illustrated in Fig. 5. Specifically, we observed the emergence of diffused and band units, as predicted by our theoretical framework in Section 3. Interestingly, we did not observe the attractive distortions predicted in Fig. 3.

An important aspect of these distortions is that they are global, whereas the reward was localized at the origin. Given that place cells average the input from multiple grid cells through their readouts, it is possible that global distortions in the grid cells facilitate local changes in the final output. For example, the ring units may arise because the reward information is rotationally symmetric, eliminating the need to retain angle information and creating the secondary ring structures shown in Fig. 5A. Additionally, if the center of the environment needs to be represented preferentially over the rest of the space, it makes sense for the representation of places outside the origin to become less crisp to conserve, *e.g.*, a firing rate budget. This could be achieved by a convolution in the firing rates, leading to the diffused units observed in Fig. 5B. Finally, grid cells can be described as a summation of a few band cells in different orientations. If these bands cover all angles, the result can approximate a place cell in the origin, making the band units, observed in Fig. 5C, a mutually synergistic solution for both representing spatial navigation and the reward information (located at the origin).

As the final test, we focused on modules that showed increased levels of diffused units and tested the reduction hypothesis presented in Conjecture 1, which suggests that the emergence of diffused units should decrease the overall size of the manifold. As expected, the diffusing units did not change the topology of the module, and resulted in a geometrical reduction in size (Fig. 6).

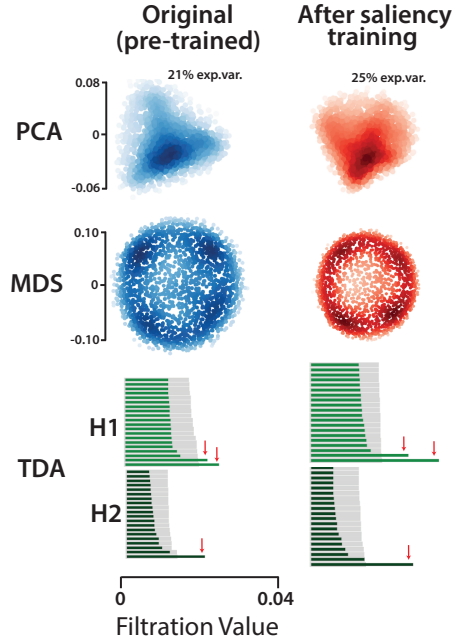

Figure 6: **Diffused units lead to reduction in the manifold size.** When we analyzed a module with many emergent diffused units after the saliency training, we observed that the topology of the module was preserved. However, the size of the manifold was reduced in line with our Conjecture 1.

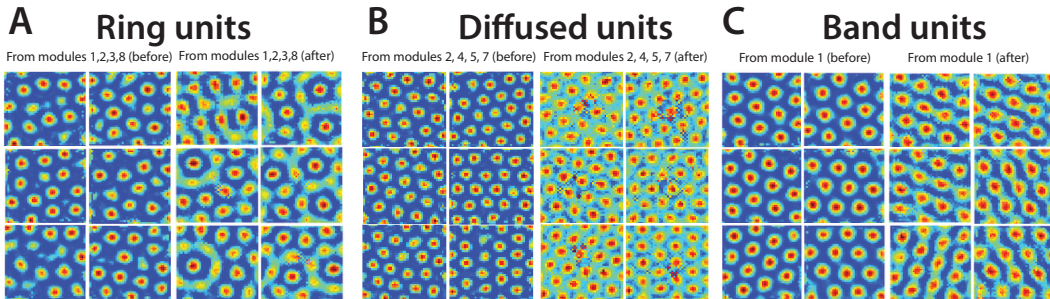

Figure 5: **Saliency-tuned piRNNs develop diverse set of tuning features observed in neuroscience. A.** Across multiple modules, we observe the emergence of many units characterized by *rings* of high activity. **B.** We also observe the emergence of *diffused* units which are active almost everywhere. **C.** Finally, we find many units develop *band* structures.

Overall, our results suggest that the global distortion mechanisms allow the grid cells to retain their spatial navigation capabilities while adapting to the localized reward signals. This balance can enable the piRNNs to maintain navigation functionality and incorporate new environmental cues effectively.

# 5 Discussion

**Local reward leads to global changes in the firing rates:** When we trained the piRNNs with local rewards, encouraging the network to represent specific places more accurately, we observed distinct global changes in the firing profiles of grid cells. In our experiments with piRNNs, we observed diffused and banded patterns, as well as units with secondary local ring-like structures similar to those observed in human studies [35], but not the attracted firing patterns.

**No attracted neurons emerged in reward-modulated piRNNs:** To our initial surprise, we did not observe the kind of local "magnification" or deformations suggested by the interepretations of previous rodent experiments from [15] in any of our experiments. Instead, we observed global changes in the firing fields. Therefore, we present *in silico* evidence that challenges the conventional interpretation that reward modulation results in local magnification of the grid cell lattice as too simplistic an explanation. Indeed, because grid cells provide a global code for space, we propose that reward modulation should result in global deformations of spatial responses in MEC, as observed in our piRNN experiments.

The appearance of diffused units suggests that multiplying the loss function by a local reward signal in the navigation space causes convolution in grid cell firing patterns, indicating a duality between grid cell firing patterns and changes in the reward function, akin to the Fourier Transform. The emergence of banded units aligns with representations of the origin, as the Fourier transform of a square wave is a periodic function with most of its mass centered at the origin. The ring units' emergence is also consistent with the reward being centered, leading to a trade-off in representing angles for better distance representation to the reward. Notably, local changes in the reward function caused global changes in grid cell firing patterns, reinforcing the concept of this duality.

**Dynamical interactions lead to distorted units:** Previous studies have shown that units with banded firing rates emerge during reorganization [36]. Additionally, research has demonstrated that banded units and grid cells can coexist, with the same cells behaving as either depending on the environment or the task [37]. A common finding across these studies, which is consistent with our results here, is that grid cells transform into banded or other unit types with different symmetries, likely influenced by the complexity of the action rather than the properties of the navigated space.

**Distorted grid cells emerge with frozen readouts, otherwise complex reorganization ensues:** The fact that the toroidal topological structure is destroyed when readouts are allowed to change supports our hypothesis that the distortions observed in grid cell firing fields are related to the network's efforts to adapt to a changing environment, such as the introduction of a reward, without completely reorganizing itself. Complete reorganization might minimize a loss function associated with the reward's location but would fail to generalize to changing environmental conditions, such as the introduction of subsequent phases of training with different saliency distributions.

We conjecture that the distortions observed in our experiments, and regularly reported in neural data [37, 36, 35, 15], signal continual learning. These distortions are expected to vary from environment to environment and task to task. This suggests that grid cells can exhibit significant flexibility, dynamically adjusting to new tasks and conditions while maintaining overall functionality. Hence, although their toroidal neural manifold topology supports navigation, global distortions may facilitate the dynamic encoding of localized environmental cues.

# 6 Conclusion

Prior work with grid cells has primarily focused on their contribution to spatial navigation. Recent studies have also considered local changes in grid cell firing patterns, such as attractions to reward positions [15]. In this work, by analyzing how piRNNs handle these tasks simultaneously, we tested whether global distortions emerge on grid cell firing patterns and how they affect their functionality and their adaptability in representing complex environmental cues. Overall, our theoretical and experimental findings suggested a nuanced view: the hexagonal grid structures facilitate the decoding of spatial locations, while global, not necessarily local, distortions in the grid cell geometry can

dynamically encode environmental cues relevant to specific actions, highlighting the flexibility of grid cells in adapting to various environmental contexts and tasks [37].

Though grid cells were not traditionally considered in the context of continual learning, our results implicate their involvement in these paradigms. Additionally, studies of spatially structured, periodic, and global rewards should illuminate the duality in the dynamic representation of environmental cues by grid cells, which may allow building towards a theoretical understanding of the duality between local rewards and global distortions we observed in this work.

## Acknowledgments and Disclosure of Funding

We thank Simon Mataigne, Adele Myers, and members of the Geometric Intelligence lab for helpful discussions and feedback on this manuscript. F. A. and N. M. acknowledge funding from the NSF grant 2313150. WTR acknowledges funding by an Internal Research and Development grant from JHU/APL. FD receives funding from Stanford University's Mind, Brain, Computation and Technology program, which is supported by the Stanford Wu Tsai Neuroscience Institute.

## References

[1] Juan A. Gallego, Matthew G. Perich, Raeed H. Chowdhury, Sara A. Solla, and Lee E. Miller. Long-term stability of cortical population dynamics underlying consistent behavior. *Nature Neuroscience*, 23(2):260–270, February 2020.

[2] Mark M. Churchland, John P. Cunningham, Matthew T. Kaufman, Justin D. Foster, Paul Nuyujukian, Stephen I. Ryu, and Krishna V. Shenoy. Neural population dynamics during reaching. *Nature*, 487(7405):51–56, July 2012.

[3] Edward H. Nieh, Manuel Schottdorf, Nicolas W. Freeman, Ryan J. Low, Sam Lewallen, Sue Ann Koay, Lucas Pinto, Jeffrey L. Gauthier, Carlos D. Brody, and David W. Tank. Geometry of abstract learned knowledge in the hippocampus. *Nature*, 595(7865):80–84, July 2021.

[4] Wei Guo, Jie J. Zhang, Jonathan P. Newman, and Matthew A. Wilson. Latent learning drives sleep-dependent plasticity in distinct CA1 subpopulations. Preprint, Neuroscience, February 2020.

[5] Gurjeet Singh, Facundo Memoli, Tigran Ishkhanov, Guillermo Sapiro, Gunnar Carlsson, and Dario L. Ringach. Topological analysis of population activity in visual cortex. *Journal of Vision*, 8(8):11, June 2008.

[6] Dean A. Pospisil and Jonathan W. Pillow. Revisiting the high-dimensional geometry of population responses in visual cortex, February 2024.

[7] Adrien Peyrache, Marie M. Lacroix, Peter C. Petersen, and György Buzsáki. Internally organized mechanisms of the head direction sense. *Nature Neuroscience*, 18(4):569–575, April 2015.

[8] Rishidev Chaudhuri, Berk Gerçek, Biraj Pandey, Adrien Peyrache, and Ila Fiete. The intrinsic attractor manifold and population dynamics of a canonical cognitive circuit across waking and sleep. *Nature Neuroscience*, 22(9):1512–1520, September 2019.

[9] Richard J. Gardner, Erik Hermansen, Marius Pachitariu, Yoram Burak, Nils A. Baas, Benjamin A. Dunn, May-Britt Moser, and Edvard I. Moser. Toroidal topology of population activity in grid cells. *Nature*, 602(7895):123–128, February 2022.

[10] Mehrdad Jazayeri and Srdjan Ostojic. Interpreting neural computations by examining intrinsic and embedding dimensionality of neural activity. *Current Opinion in Neurobiology*, 70:113–120, October 2021.

[11] Peiran Gao, Eric Trautmann, Byron Yu, Gopal Santhanam, Stephen Ryu, Krishna Shenoy, and Surya Ganguli. A theory of multineuronal dimensionality, dynamics and measurement. Preprint, Neuroscience, November 2017.

[12] SueYeon Chung and L. F. Abbott. Neural population geometry: An approach for understanding biological and artificial neural networks. *Current Opinion in Neurobiology*, 70:137–144, October 2021.

[13] Torkel Hafting, Marianne Fyhn, Sturla Molden, May-Britt Moser, and Edvard I. Moser. Microstructure of a spatial map in the entorhinal cortex. *Nature*, 436(7052):801–806, August 2005.

[14] Hanne Stensola, Tor Stensola, Trygve Solstad, Kristian Frøland, May-Britt Moser, and Edvard I. Moser. The entorhinal grid map is discretized. *Nature*, 492(7427):72–78, December 2012.

[15] Charlotte N. Boccara, Michele Nardin, Federico Stella, Joseph O'Neill, and Jozsef Csicsvari. The entorhinal cognitive map is attracted to goals. *Science*, 363(6434):1443–1447, March 2019.

[16] William N. Butler, Kiah Hardcastle, and Lisa M. Giocomo. Remembered Reward Locations Restructure Entorhinal Spatial Maps. *Science (New York, N.Y.)*, 363(6434):1447–1452, March 2019.

[17] Julija Krupic, Marius Bauza, Stephen Burton, and John O'Keefe. Local transformations of the hippocampal cognitive map. *Science*, 359(6380):1143–1146, March 2018.

[18] Gily Ginosar, Johnatan Aljadeff, Liora Las, Dori Derdikman, and Nachum Ulanovsky. Are grid cells used for navigation? On local metrics, subjective spaces, and black holes. *Neuron*, pages S0896–6273(23)00223–4, April 2023.

[19] Dehong Xu, Ruiqi Gao, Wen-Hao Zhang, Xue-Xin Wei, and Ying Nian Wu. Conformal Isometry of Lie Group Representation in Recurrent Network of Grid Cells, November 2022.

[20] Erik Hermansen, David A. Klindt, and Benjamin A. Dunn. Uncovering 2-D toroidal representations in grid cell ensemble activity during 1-D behavior, November 2022.

[21] Alexei Samsonovich and Bruce L. McNaughton. Path Integration and Cognitive Mapping in a Continuous Attractor Neural Network Model. *Journal of Neuroscience*, 17(15):5900–5920, August 1997.

[22] Yoram Burak and Ila R. Fiete. Accurate Path Integration in Continuous Attractor Network Models of Grid Cells. *PLoS Computational Biology*, 5:e1000291, February 2009.

[23] J. O'Keefe and J. Dostrovsky. The hippocampus as a spatial map. Preliminary evidence from unit activity in the freely-moving rat. *Brain Research*, 34(1):171–175, November 1971.

[24] John O'Keefe and Lynn Nadel. *The Hippocampus as a Cognitive Map*. Oxford University Press, 1978.

[25] Marianne Fyhn, Torkel Hafting, Alessandro Treves, May-Britt Moser, and Edvard I Moser. Hippocampal remapping and grid realignment in entorhinal cortex. *Nature*, 446(7132):190–194, 2007.

[26] KiJung Yoon, Michael A Buice, Caswell Barry, Robin Hayman, Neil Burgess, and Ila R Fiete. Specific evidence of low-dimensional continuous attractor dynamics in grid cells. *Nature neuroscience*, 16(8):1077–1084, 2013.

[27] Mariana Gil, Mihai Ancau, Magdalene I Schlesiger, Angela Neitz, Kevin Allen, Rodrigo J De Marco, and Hannah Monyer. Impaired path integration in mice with disrupted grid cell firing. *Nature neuroscience*, 21(1):81–91, 2018.

[28] Christopher J. Cueva and Xue-Xin Wei. Emergence of grid-like representations by training recurrent neural networks to perform spatial localization, March 2018.

[29] Andrea Banino, Caswell Barry, Benigno Uria, Charles Blundell, Timothy Lillicrap, Piotr Mirowski, Alexander Pritzel, Martin J. Chadwick, Thomas Degris, Joseph Modayil, Greg Wayne, Hubert Soyer, Fabio Viola, Brian Zhang, Ross Goroshin, Neil Rabinowitz, Razvan Pascanu, Charlie Beattie, Stig Petersen, Amir Sadik, Stephen Gaffney, Helen King, Koray Kavukcuoglu, Demis Hassabis, Raia Hadsell, and Dharshan Kumaran. Vector-based navigation using grid-like representations in artificial agents. *Nature*, 557(7705):429–433, May 2018.

[30] Ruiqi Gao, Jianwen Xie, Xue-Xin Wei, Song-Chun Zhu, and Ying Nian Wu. On Path Integration of Grid Cells: Group Representation and Isotropic Scaling, November 2021.

[31] Ben Sorscher, Gabriel C. Mel, Samuel A. Ocko, Lisa M. Giocomo, and Surya Ganguli. A unified theory for the computational and mechanistic origins of grid cells. *Neuron*, 111(1):121–137.e13, January 2023.

[32] Rylan Schaeffer, Mikail Khona, Tzuhsuan Ma, Cristóbal Eyzaguirre, Sanmi Koyejo, and Ila Rani Fiete. Self-Supervised Learning of Representations for Space Generates Multi-Modular Grid Cells, November 2023.

[33] Markus Borud Pettersen, Vemund Sigmundson Schøyen, Anders Malthe-Sørenssen, and Mikkel Elle Lepperød. Decoding the Cognitive map: Learning place cells and remapping, March 2024.

[34] Rosamund F. Langston, James A. Ainge, Jonathan J. Couey, Cathrin B. Canto, Tale L. Bjerknes, Menno P. Witter, Edvard I. Moser, and May-Britt Moser. Development of the Spatial Representation System in the Rat. *Science*, 328(5985):1576–1580, June 2010.

[35] Joshua Jacobs, Christoph T Weidemann, Jonathan F Miller, Alec Solway, John F Burke, Xue-Xin Wei, Nanthia Suthana, Michael R Sperling, Ashwini D Sharan, Itzhak Fried, et al. Direct recordings of grid-like neuronal activity in human spatial navigation. *Nature neuroscience*, 16(9):1188–1190, 2013.

[36] Vemund Schøyen, Markus Borud Pettersen, Konstantin Holzhausen, Anders Malthe-Sørenssen, and Mikkel Elle Lepperød. Coherently Remapping Toroidal Cells But Not Grid Cells are Responsible for Path Integration in Virtual Agents, August 2022.

[37] Julija Krupic, Neil Burgess, and John O'Keefe. Neural representations of location composed of spatially periodic bands. *Science (New York, N.Y.)*, 337(6096):853–857, August 2012.

[38] Leland McInnes, John Healy, and James Melville. UMAP: Uniform Manifold Approximation and Projection for Dimension Reduction, September 2020.

[39] Martin Ester, Hans-Peter Kriegel, Jörg Sander, and Xiaowei Xu. A density-based algorithm for discovering clusters in large spatial databases with noise. In *Proceedings of the Second International Conference on Knowledge Discovery and Data Mining*, KDD'96, pages 226–231, Portland, Oregon, August 1996. AAAI Press.

[40] Francisco Acosta, Sophia Sanborn, Khanh Dao Duc, Manu Madhav, and Nina Miolane. Quantifying Extrinsic Curvature in Neural Manifolds. In *Proceedings of the IEEE/CVF Conference on Computer Vision and Pattern Recognition*, pages 610–619, 2023.

[41] Chad Giusti, Eva Pastalkova, Carina Curto, and Vladimir Itskov. Clique topology reveals intrinsic geometric structure in neural correlations. *Proceedings of the National Academy of Sciences*, 112(44):13455–13460, November 2015.

[42] Gard Spreemann, Benjamin Dunn, Magnus Bakke Botnan, and Nils A. Baas. Using persistent homology to reveal hidden information in neural data, October 2015.

[43] Julián Burella Pérez, Sydney Hauke, Umberto Lupo, Matteo Caorsi, and Alberto Dassatti. Giotto-ph: A Python Library for High-Performance Computation of Persistent Homology of Vietoris-Rips Filtrations, August 2021.

# S1 Proofs of Results from the Theory Section

Here we provide a more detailed discussion and derivations of the theory introduced in Section 3.

## S1.1 Diffusion of rate maps

We start by proving the following proposition, about a single grid cell $i$. This proposition refers to the definition of $r_i^\Phi$ from Eq. 4.

**Proposition 1.** *The deformation of the grid cell firing fields $r_i$ by $\Phi$ given by $r_i * \Phi = r_i^\Phi$ defines a right group action of the group of diffeomorphisms over the space of firing fields. The firing energy budget is an invariant of this group action.*

*Proof.* We first prove that $*$ defines a valid right group action, by proving that it verifies the group action's axioms: identity and compatibility.

The identity diffeomorphism $\Phi = \mathrm{id}$ maps every point to itself $\Phi(x) = x$, has its Jacobian equal to the identity matrix with determinant 1. Thus, $r_i^{\mathrm{id}}(x) = |\det \mathrm{id}'(x)| r_i \circ \mathrm{id}(x) = |1| r_i(x) = r_i(x)$. This shows that the scalar field remains unchanged under the identity diffeomorphism, satisfying the identity axiom of a group action.

The composition of two diffeomorphisms $\Phi$ and $\Psi$ acts on the firing fields as:

$$r_i^{\Psi \circ \Phi}(x) = |\det(\Psi \circ \Phi)'(x)| r_i \circ (\Psi \circ \Phi)(x)$$

By the chain rule for determinants of Jacobians, we get:

$$r_i^{\Psi \circ \Phi}(x) = |\det \Psi'(\Phi(x))| \cdot |\det \Phi'(x)| r_i \circ (\Psi \circ \Phi)(x).$$

This expression matches the successive application of the transformations:

$$\begin{aligned}
(r_i * \Psi) * \Phi(x) &= |\det \Phi'(x)| (r_i * \Psi)(\Phi(x)) \\
&= |\det \Phi'(x)| |\det \Psi'(\Phi(x))| r_i \circ \Psi(\Phi(x)) \\
&= r_i * (\Psi \circ \Phi)(x)
\end{aligned}$$

This shows the compatibility property for a right group action. Further, the firing energy budget is an invariant of the group action. Indeed, by definition of a multivariate change of variables, we have

$$\begin{aligned}
\int_{x \in \mathbb{R}^2} r_i^\Phi(x) dx &= \int_{x \in \mathbb{R}^2} |\det \Phi'(x)| r_i \circ \Phi(x) dx \\
&= \int_{\Phi(\mathbb{R}^2)} r_i(x) dx \\
&= \int_{x \in \mathbb{R}^2} r_i(x) dx
\end{aligned}$$

for every grid cell $i$. $\qquad\qquad\square$

Next, we prove the result regarding the deformation of the neural manifold.

**Proposition 2.** *The deformation of the neural manifold $R$ by $\Phi$ given by $\Phi * R = R^\Phi$ defines a right group action of the group of diffeomorphisms over the space of neural manifolds. The barycenter and the topology of the manifold are invariants of this group action, provided that the firing fields are regular enough.*

*Proof.* We first prove that $*$ defines a valid right group action, by proving that it verifies the group action's axioms: identity and right compatibility. These axioms are verified, using the same arguments as the ones used in the proof of group action over the firing fields above.

Next, we prove that the barycenter of the neural manifold is unchanged by the deformation. The barycenter before deformation is:

$$\int_{x \in \mathbb{R}^2} R(x) dx, \qquad\qquad (S1)$$

and, after deformation:

$$\int_{x \in \mathbb{R}^2} R^\Phi(x)dx = \int_{x \in \mathbb{R}^2} |\det \Phi'(x)| . R \circ \Phi(x)dx = \int_{x \in \mathbb{R}^2} R(x)dx, \qquad \text{(S2)}$$

by definition of the multivariate change of variables. Thus, the barycenter is unchanged.

Next, we show that the topology of the manifold $M^\Phi$ defined by $R^\Phi$ is the same as the topology of the manifold $M$ defined by $R$.

Two manifolds are topologically equivalent (homeomorphic) if there exists an homeomorphism $\Psi : M \to M^\Phi$ between them, i.e., a map $\Psi$ that is continuous, bijective, with a continuous inverse.

Consider the map $\Psi : M \to M^\Phi$, defined for $p = R(x) \in M$, where $x$ is the unique pre-image of $p$ by injectivity, as:

$$\Psi(p) = r^\Phi(x) = |\det \Phi'(x)| r \circ \Phi(x) = |\det \Phi'(R^{-1}(p))| \Phi(R^{-1}(p)).$$

We show that $\Psi$ is continuous. The diffeomorphism $\Phi$ is smooth on $\mathbb{R}^2$ by definition. The function $R$ is also smooth since every firing field is smooth. Next, $R^{-1}$ is also considered smooth, so that $\Phi \circ R^{-1}$ is smooth. Thus, the map $|\det \Phi'(R(x))|$ is smooth as it is the determinant of a smooth function. Thus, $\Psi$ is smooth and therefore continuous.

Further, $\Psi$ is bijective: $\Phi$ and $R^{-1}$ are bijective by definition of a diffeomorphism and because we restricted $R$ to its domain of injectivity. The determinant is non 0 for the same reason. Thus, $\Psi$ is bijective. We assume that $R$ is regular enough so that the inverse of $\Psi$ is continuous as well. Thus, $\Psi$ is an homeomorphism and the two manifolds are topologically equivalent. $\qquad \square$

## S2 Refinement on theory following rebuttal

### S2.1 Diffused Units: Isotropic Smoothing of Firing Fields

We show how diffused units, observed in practice in Section 4, lead to smaller neural manifolds in neural state space.

To this aim, we represent the firing field $r$ of one grid cell as a mixture of $K$ Gaussians, centered at locations $\mu_1, ..., \mu_K$ that are regularly distributed along a grid. That is, for $x \in \mathbb{R}^2$:

$$r(x) = \frac{1}{K} \sum_{k=1}^K G(\mu_k, \sigma_r^2)(x), \quad \text{where: } G(\mu_k, \sigma_r^2)(x) = \frac{1}{2\pi\sigma_r^2} \exp\left(-\frac{\|x - \mu_k\|^2}{2\sigma_r^2}\right). \qquad \text{(S3)}$$

Next, we describe the transformation of the grid cell units into diffused units as a convolution with a 2D isotropic Gaussian filter $G(0, \sigma^2)$ for some variance $\sigma^2$. Consequently, instead of writing this transformation as the action of a diffeomorphism $\Phi$, we write is as the action of $\sigma \in \mathbb{R}_+^*$.

$$r^\sigma(x) = \left(G(0, \sigma^2) * r\right)(x), \ \forall x \in \mathbb{R}^2. \qquad \text{(S4)}$$

We now prove our main result on diffused units.

**Proposition 3.** *When the original grid cell units become diffused units after integration of a reward: (i) the energy budget is unchanged, and (ii) the size of the original neural manifold decreases, where the size is defined as $S = \max_{x \in \mathbb{R}^2} \|R(x)\|_\infty$, with $R(x) = (r_1(x), ..., r_N(x))$ for $N$ units.*

*Proof.* (i). We first prove that the energy budget is unchanged.

The energy budget before the introduction of reward, for a original unit, is:

$$E = \int_{x \in \mathbb{R}^2} r(x) dx$$

$$= \frac{1}{K} \int_{x \in \mathbb{R}^2} \sum_{k=1}^{K} G(\mu_k, \sigma_r^2)(x) dx$$

$$= \frac{1}{K} \sum_{k=1}^{K} \int_{x \in \mathbb{R}^2} G(\mu_k, \sigma_r^2)(x) dx$$

$$= \frac{1}{K} \sum_{k=1}^{K} 1 dx$$

$$= 1.$$

To compute the energy budget after the introduction of reward, for a diffused unit, we first provide an alternative formula for $r^\sigma(x)$. We have:

$$r^\sigma(x) = \left( G(0, \sigma^2) * r \right)(x)$$

$$= \left( G(0, \sigma^2) * \frac{1}{K} \sum_{k=1}^{K} G(\mu_k, \sigma_r^2) \right)(x)$$

$$= \frac{1}{K} \sum_{k=1}^{K} \left( G(0, \sigma^2) * G(\mu_k, \sigma_r^2) \right)(x)$$

$$= \frac{1}{K} \sum_{k=1}^{K} G(\mu_k, \sigma^2 + \sigma_r^2)(x),$$

where we have used the fact that the convolution is a linear operation, and the fact that the convolution of two Gaussians is a Gaussian with added means and variances.

The energy budget after the introduction of reward, for a diffused unit, is therefore:

$$E^\sigma = \int_{x \in \mathbb{R}^2} r^\sigma(x) dx$$

$$= \int_{x \in \mathbb{R}^2} \frac{1}{K} \sum_{k=1}^{K} G(\mu_k, \sigma^2 + \sigma_r^2)(x) dx$$

$$= 1.$$

Therefore, the energy budget is conserved: $E = E^\sigma$.

(ii). Next, we prove that the size of the neural manifold corresponding to the diffused units is smaller than the size of the neural manifolds of the original units.

Consider a neural manifold corresponding to $N$ units. The size of the manifold is the maximum value taken by any of its $N$ units, across the domain $\mathbb{R}^2$.

$$S = \max_{x \in \mathbb{R}^2} \|R(x)\|_\infty = \max_{x \in \mathbb{R}^2} \max_{i \in N} r_i(x). \tag{S5}$$

Since all units have the same profile, their maxima are equal. Hence, we compute the maximum of one original unit and of one diffused unit. Additionally, each unit is a mixture of $K$ Gaussians. Assuming that the Gaussians are disjoints enough, which is the case in practice, we compute the maximum of a unit as the maximum of one of its Gaussians.

Accordingly, the size of the neural manifold corresponding to the original units is:

$$S = \max_{x \in \mathbb{R}^2} r(x) = \frac{1}{K} \max_{x \in \mathbb{R}^2} G(\mu_1, \sigma_r^2) = \frac{1}{K} \frac{1}{2\pi \sigma_r^2}.$$

Next, the size of the neural manifold corresponding to the diffused units is:

$$S^\sigma = \max_{x \in \mathbb{R}^2} r(x) = \frac{1}{K} \max_{x \in \mathbb{R}^2} G(\mu_1, \sigma^2 + \sigma_r^2) = \frac{1}{K} \frac{1}{2\pi (\sigma^2 + \sigma_r^2)}.$$

Thus, we have $S^\sigma < S$, which proves that the neural manifold has a smaller size. □

## S2.2 Band Units: Anisotropic Smoothing of Firing Fields

We show how band units, observed in practice in Section 4, lead to a torus manifold that collapses into a circle manifold. As in the previous subsection, we represent the firing field $r$ of one unit as a mixture of $K$ Gaussians as in Eq. S3.

Next, we describe the transformation of units into band units as a convolution with a 2D anisotropic Gaussian filter $G(0, \Sigma)$ for $\Sigma \in \text{Sym}(2)$ with eigenvalues $(0, \sigma^2)$ with $\sigma^2 >> 1$.

Consequently, instead of writing this transformation as the action of a diffeomorphism $\Phi$, we write is as the action of $\Sigma \in \text{Sym}(2)$.

$$r^\sigma(x) = (G(0, \Sigma) * r)(x), \ \forall x \in \mathbb{R}^2. \tag{S6}$$

We now prove our main result on band units.

**Proposition 4.** *When the original units become band units after integration of a reward: (i) the energy budget is unchanged, and (ii) the original neural manifold (a torus) collapses into a circle.*

*Proof.* 1. We first prove that the energy budget is unchanged.

We recall from the previous subsection that the energy budget before the introduction of reward, for a original unit, is:

$$E = \int_{x \in \mathbb{R}^2} r(x)dx = 1.$$

To compute the energy budget after the introduction of reward, for a band unit, we first provide an alternative formula for $r^\sigma(x)$. We have:

$$
\begin{aligned}
r^\Sigma(x) &= (G(0, \Sigma) * r)(x) \\
&= \left( G(0, \Sigma) * \frac{1}{K} \sum_{k=1}^{K} G(\mu_k, \sigma_r^2) \right)(x) \\
&= \frac{1}{K} \sum_{k=1}^{K} \left( G(0, \Sigma) * G(\mu_k, \sigma_r^2) \right)(x) \\
&= \frac{1}{K} \sum_{k=1}^{K} G(\mu_k, \Sigma + \sigma_r^2.I)(x).
\end{aligned}
$$

where we have used the fact that the convolution is a linear operation, and the fact that the convolution of two Gaussians is a Gaussian with added means and variances.

The energy budget after the introduction of reward, for a diffused unit, is:

$$
\begin{aligned}
E^\Sigma &= \int_{x \in \mathbb{R}^2} r^\sigma(x)dx \\
&= \int_{x \in \mathbb{R}^2} \frac{1}{K} \sum_{k=1}^{K} G(\mu_k, \Sigma + \sigma_r^2.I)(x)dx \\
&= 1.
\end{aligned}
$$

Therefore, the energy budget is conserved: $E = E^\Sigma$.

2. Next, we prove that neural manifold evolves from a torus to a circle.

We do this by proving that the application of a Gaussian filter is a continuous operation in the variable $\sigma^2$. For this, we use the spectral theorem, which states that every real, symmetric matrix is diagonalizable in an orthonormal basis, to write $\Sigma$ as:

$$\Sigma = R\text{diag}(0, \sigma^2)R^{-1}, \tag{S7}$$

where $R$ is a 2D rotation matrix. Accordingly, we see that $\sigma$ impacts the firing field $r$ continuously, as follows:

$$r^\Sigma(x) = \frac{1}{K} \sum_{k=1}^{K} G(\mu_k, R\text{diag}(\sigma_r^2, \sigma^2 + \sigma_r^2)R^{-1})(x).$$

Thus, the evolution of the neural manifold as $\sigma$ increases is continuous. At the limit, where $\sigma^2 \to \infty$ we have the band units observed in practice.

Next, we prove using group theory that band units generate a circle manifold. (TODO: Add theory written by Fran).

Consequently, the progression from the original units to the band units is a progression from a torus manifold to a circle manifold.

$\square$

### S2.3    Ring Units: Angular Smoothing of Firing Fields

We show how ring units, observed in practice in Section 4, can be described via an angular smoothing of firing fields. As in the previous subsection, we represent the firing field $r$ of one unit as a mixture of $K$ Gaussians as in Eq. S3.

Next, we describe the transformation of units into ring units as a convolution with a 2D anisotropic *variable* Gaussian filter. In this convolution, the kernel varies depending on where it is being applied, i.e., on its position which we will express in polar coordinates with $(\rho, \theta)$.

In other words, at $(\rho, \theta)$ fixed, the function $u \to G_{\text{var}}(u; \rho, \theta)$ is a Gaussian kernel with a mean and variance that may depend on $\rho$ and $\theta$. In our case, given a reference radius $\rho_0 > 0$, the dependency of the mean and variable on $(\rho, \theta)$ has the form:

$$G_{\text{var}}(u; \rho, \theta) = \delta_{\rho=\rho_0}.G(0, \Sigma_\theta)(u) = \delta_{\rho=\rho_0}.\frac{1}{2\pi\sqrt{\det \Sigma_\theta}} \exp\left(-\frac{1}{2}u^T.\Sigma_\theta^{-1}.u\right), \ \forall u \in \mathbb{R}^2, \quad \text{(S8)}$$

where $\delta$ is the Kronecker symbol and $\Sigma_\theta = R_\theta.\text{diag}(0, \sigma^2).R_\theta^{-1}$. The mean of the Gaussian kernel is always 0, but the covariance matrix depends on $\theta$. The Gaussian kernel itself is 0 if $\rho$ is different than the reference radius $\rho_0$.

This variable kernel can be used to perform an anisotropic smoothing in the direction $\theta$ for a ring of radius $\rho_0$, via convolution. For simplicity, we consider that the center of the ring is also the center of the polar coordinate system. We write the convolution with this variable kernel as:

$$r^{\rho_0}(x) = \left(G_{\text{var}} * r\right)(x), = \int_{u \in \mathbb{R}^2} r(u)G_{\text{var}}(x - u; x)du = \int_{u \in \mathbb{R}^2} r(u)G_{\text{var}}(x - u; \rho, \theta)du, \quad \text{(S9)}$$

where we write $x$ in polar coordinates $x = (\rho, \theta)$.

In Figure S1, we verify empirically that this formulation creates the rings that we observe in practice.

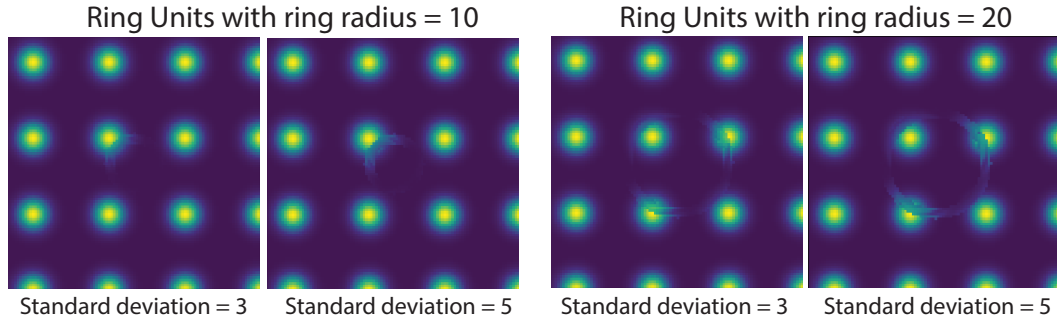

Figure S1: Convolution with a variable Gaussian kernel creates the ring units observed in practice. From left to right, we vary the radius $\rho_0$ and the standard deviation $\sigma$ of the variable Gaussian kernel.

## S3 Network details & hyperparameters

All code used for this paper is publicly available at:
`https://github.com/geometric-intelligence/neurometry`.

**Grid cells in piRNNs** Gao et al. [30] and Xu et al. [19] introduce a framework where transformations in the environment are related to transformations in neural state space. That is, for a displacement $\Delta x$, the neural representations $r$ transform as:

$$r(x + \Delta x) = F_{\Delta x}(r(x)), \tag{S10}$$

where $F$ is assumed to respect *conformal isometry*—a small displacement vector $\Delta x$ at $x$ in physical space is conformally mapped to a small displacement vector $r(x + \Delta x)$ in neural state space. This can be understood as a "magnification" of the displacement by a constant factor $s(x)$ that may depend on $x$, but is independent of the direction of $\Delta x$. Xu et al. [19] approximate the transformation $F_{\Delta x}(r(x))$ using an RNN:

$$r(x + \Delta x) = \text{ReLU}(Wr(x) + U\Delta x), \tag{S11}$$

where $W \in \mathbb{R}^{N \times N}$ is the recurrent weight matrix and $U \in \mathbb{R}^{N \times 2}$ is the input weight matrix with $\Delta x$ called the input. The neural representations $r(x)$ will become grid cells after training. To incorporate the fact that biological grid cells are organized into discrete modules [14], they assume that the representations $r(x)$ are divided into sub-vectors corresponding to modules, and therefore they assume that $W$ is a block-diagonal matrix and $U$ is divided into sub-blocks by row.

Next, we provide expressions for the remaining loss terms from Eq. 2. The kernel term is:

$$\mathcal{L}_{\text{kernel}} = \mathbb{E}_x \left[ \|\phi(x) - Qr(x)\|^2 \right] \tag{S12}$$

Since $\phi$ is fixed, this term induces the linear read-out of grid cell representations $r$ with matrix $Q$ to match the Gaussian kernel.

The conformal isometry term is:

$$\mathcal{L}_{\text{conformal}} = \mathbb{E}_{x, \Delta x_1, \Delta x_2} \left[ s_{\Delta x_1}(x) - s_{\Delta x_2}(x) \right]^2, \tag{S13}$$

where $s_{\Delta x}(x) = (\|r(x) - r(x + \Delta x)\| / \|\Delta x\|)^2$. This term enforces that the magnitude of displacements in neural state space induced by a displacement in the real environment is independent of the direction of the displacement at each point. Thus, the grid cell representation taken as a map from $\mathbb{R}^2$ to $\mathbb{R}^N$ is conformal (angles are preserved locally).

The regularization term is:

$$\mathcal{L}_{\text{regularization}} = \|Q\|_F^2. \tag{S14}$$

This term can be interpreted as inducing sparsity in the place cell activity.

### S3.1 Pre-training Details

We pre-trained the model from [19] for 25,000 epochs with the task, architecture, and training hyperparameters originally used in their work, specified in Table 1. Every 500 epochs, we extracted the network activities and averaged over 100 trajectories.

Once the network was trained and learned units with a high degree of hexagonal periodicity across multiple modules, we modified the loss term (see Eq. 7) and initiated a new phase of training, *saliency* training. We ran 667 runs spanning different experimental conditions (see Table 2) and a hyperparamater search (see Table 3)

In our analysis in the main text we use a Gaussian saliency kernel located at the center of the environment, with $\sigma_* = 0.05$, shown in Fig. S2.

## S4 Identification of neuron subpopulations

The activity of each unit can be visualized as a (res $\times$ res) rate map, with res $= 40$. Building upon the methodologies in [9], we attempt to discover subpopulations in the network by clustering RNN units based on their spatial autocorrelations. We begin by computing the spatial autocorrelations of all 1800 RNN units, where each spatial autocorrelation is a matrix of size (res $- 1$) $\times$ (res $- 1$). We proceed by flattening the matrix, yielding 1800 points in a $\mathbb{R}^{(\text{res}-1) \times (\text{res}-1)}$, and reduce the dimension to 2D using UMAP [38]. Finally, we use DBSCAN [39] to cluster the points in the 2D UMAP projection, resulting in our final unit clusters. These clusters correspond to the usual notions of grid cell module. The 2D UMAP projection colored according to DBSCAN-obtained clusters is shown in Fig. S3.

| Architecture | |
|---|---|
| $N_{grid}$ | 1800 |
| RNN step | 10 |
| place cell $\sigma$ | 0.07 |
| Number of modules | 150 |
| Resolution (res) | 40 |
| **Training** | |
| Number of epochs | 25,000 |
| Learning rate | 0.006 |
| Optimizer | adam |
| Batch size | 10,000 |
| kernel weight | 1.05 |
| error weight | 0.1 |
| conformal weight | 0.005 |
| $Q$ reg. weight | 0.2 |
| Path length | 30 |

Table 1: Pre-trained model parameters.

| Experiments | |
|---|---|
| $N_{grid}$ | 1800 |
| RNN step | [10,20,40,60] |
| error weight | [0.1,0.3,0.5,0.9,2] |
| $s_0$ | [1,10,100,1000] |
| $x_*$ | [(0.5,0.5),(0.8,0.8)] |
| $\sigma_*$ | [0.05,0.1,0.15,0.2,0.5] |

Table 2: Different experimental conditions.

## S5  Geometric Methods in Neuroscience

There exist many approaches for revealing manifold structure in neural population activity which rely on dimensionality reduction techniques like PCA, UMAP, MDS, Isomap, LLE, and tSNE. These techniques can reveal the existence of lower-dimensional structure in neural population activity; however, these methods may misrepresent manifolds with non-trivial topological structure, like circles, spheres, and tori. Topological data analysis (TDA) methods like persistent homology hrevealed [8, 9]. However, these methods do not reveal many geometric properties of neural manifolds including notions of distance, angles, or curvature. Recent methods [40] quantify the extrinsic curvature neural manifolds leveraging tools from differential geometry.

### S5.1  Topological Data Analysis

Topological Data Analysis (TDA) has been used extensively to study the structure of neural population activity [5, 8, 9, 41, 42]. In this work we leverage the parallelized CPU computations provided by giotto-ph [43] to compute persistent homology based on Vietoris-Rips filtrations. We begin with a projection of the module activities to a 6 the dimensional space defined by PCA. We then compute the persistent homology setting the prime field $p = 2$, and Euclidean distance as the metric.

## S6  Limitations

We introduce a saliency-trained piRNN framework building off the model from [19] and [30]. Many other works propose their own versions of piRNNs with different specifications, which we have not fully explored. In particular the recent work of [32] introduce a novel self-supervised piRNN

| Experiments | |
|---|---|
| Learning rate | [3e-4, 6e-4] |
| Path length | [50,75,100] |

Table 3: Hyperparameter search for saliency training.

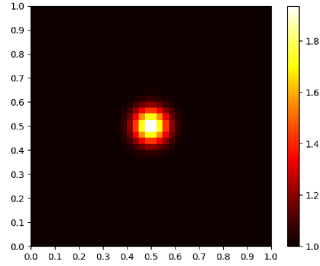

Figure S2: Saliency kernel for analysis in Sec. 4.1

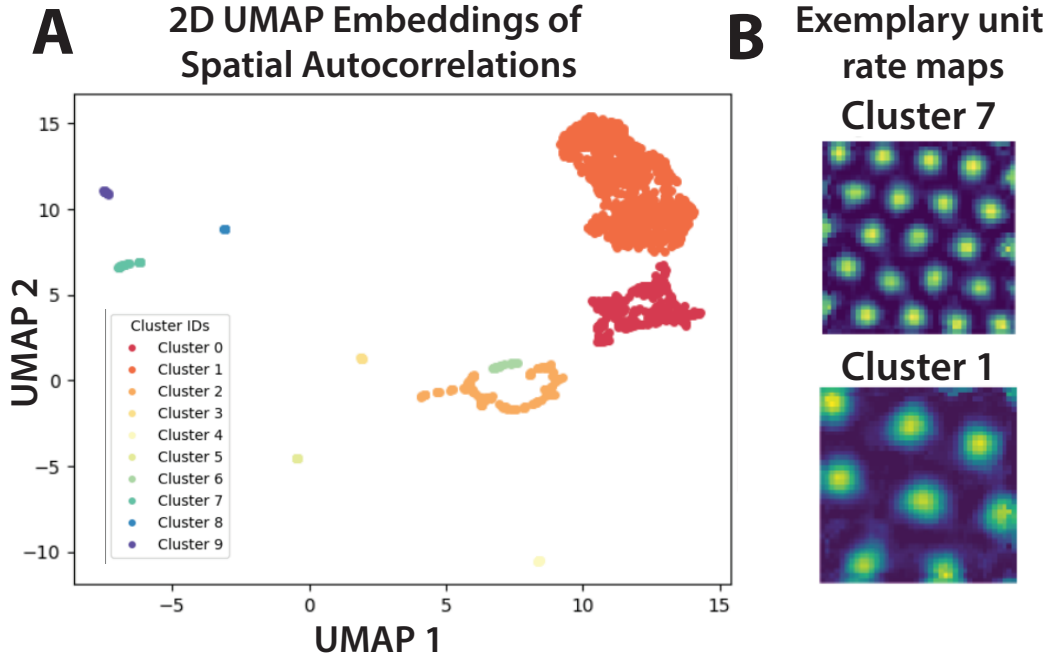

Figure S3: **Identification of network subpopulations by clustering of unit rate maps.**. **A** 2D UMAP embeddings of spatial autocorrelations of rate maps of all RNN units, colored according to cluster membership determined with DBSCAN clustering on 2D embeddings. **B** Rate maps for typical units in cluster 7 (top) and cluster 1 (bottom).

framework which considerably differs from the rest, and future work should explore the effects of saliency training on this novel type of piRNN. Furthermore, while we intend to bridge the gap between path-integrating artificial neural systems and observational evidence from experiments in navigational neuroscience, the present work does not directly analyze neural data from real animals.

## S7 Reproducibility

In training of our piRNNs and conducting our analyses, we made use of 10 NVIDIA A100 GPUs and 20 NVIDIA A100 GPUs spread over 5 clusters.

**A**                                    **B**

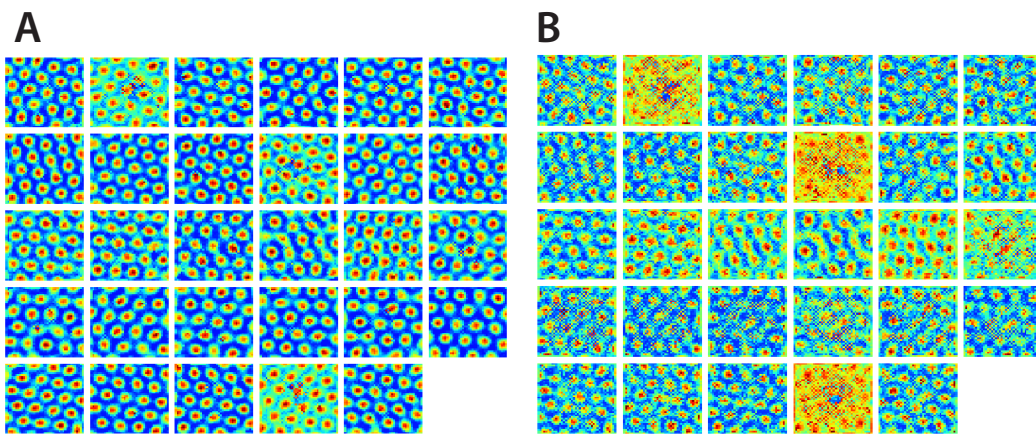

Figure S4: **A.** Rate maps for module 7 of piRNN with frozen linear read-out during saliency training. We observe the stereotypical deformations from the pure hexagonal grids described in Sec. 4, and the toroidal topology is preserved. **B.** Rate maps for module 7 of piRNN with trainable linear read-out during saliency training. The deformations are more extreme, and toroidal topology of the module is destroyed.

